# Learned Prioritization for Trading Off Accuracy and Speed[*]

**Jiarong Jiang**[*]      **Adam Teichert**[†]      **Hal Daumé III**[*]      **Jason Eisner**[†]

[*]**Department of Computer Science**
University of Maryland
College Park, MD 20742
`{jiarong,hal}@umiacs.umd.edu`

[†]**Department of Computer Science**
Johns Hopkins University
Baltimore, MD 21218
`{teichert,eisner}@jhu.edu`

## Abstract

Users want inference to be both fast and accurate, but quality often comes at the cost of speed. The field has experimented with approximate inference algorithms that make different speed-accuracy tradeoffs (for particular problems and datasets). We aim to explore this space automatically, focusing here on the case of agenda-based syntactic parsing [12]. Unfortunately, off-the-shelf reinforcement learning techniques fail to learn good policies: the state space is simply too large to explore naively. An attempt to counteract this by applying imitation learning algorithms also fails: the "teacher" follows a far better policy than anything in our learner's policy space, free of the speed-accuracy tradeoff that arises when oracle information is unavailable, and thus largely insensitive to the *known* reward functtion. We propose a hybrid reinforcement/apprenticeship learning algorithm that learns to speed up an initial policy, trading off accuracy for speed according to various settings of a speed term in the loss function.

## 1 Introduction

The nominal goal of predictive inference is to achieve high accuracy. Unfortunately, high accuracy often comes at the price of slow computation. In practice one wants a "reasonable" tradeoff between accuracy and speed. But the definition of "reasonable" varies with the application. Our goal is to optimize a system with respect to a user-specified speed/accuracy tradeoff, on a user-specified data distribution. We formalize our problem in terms of learning priority functions for generic inference algorithms (Section 2).

Much research in natural language processing (NLP) has been dedicated to finding speedups for exact or approximate computation in a wide range of inference problems including sequence tagging, constituent parsing, dependency parsing, and machine translation. Many of the speedup strategies in the literature can be expressed as pruning or prioritization heuristics. Prioritization heuristics govern the order in which search actions are taken while pruning heuristics explicitly dictate whether particular actions should be taken at all. Examples of prioritization include A$^*$ [13] and Hierarchical A$^*$ [19] heuristics, which, in the case of agenda-based parsing, prioritize parse actions so as to reduce work while maintaining the guarantee that the most likely parse is found. Alternatively, coarse-to-fine pruning [21], classifier-based pruning [23], [22] beam-width prediction [3], etc can result in even faster inference if a small amount of search error can be tolerated.

Unfortunately, deciding which techniques to use for a specific setting can be difficult: it is impractical to "try everything." In the same way that statistical learning has dramatically improved the *accuracy* of NLP applications, we seek to develop statistical learning technology that can dramatically improve their *speed* while maintaining tolerable accuracy. By combining reinforcement learning and imitation learning methods, we develop an algorithm that can successfully learn such a tradeoff in the context of constituency parsing. Although this paper focuses on parsing, we expect the approach to transfer to prioritization in other agenda-based algorithms, such as machine translation and residual belief propagation. We give a broader discussion of this setting in [8].

---

[*]This material is based upon work supported by the National Science Foundation under Grant No. 0964681.

## 2 Priority-based Inference

Inference algorithms in NLP (e.g. parsers, taggers, or translation systems) as well as more broadly in artificial intelligence (e.g., planners) often rely on prioritized exploration. For concreteness, we describe inference in the context of parsing, though it is well known that this setting captures all the essential structure of a much larger family of "deductive inference" problems [12, 9].

### 2.1 Prioritized Parsing

Given a probabilistic context-free grammar, one approach to inferring the best parse tree for a given sentence is to build the tree from the bottom up by dynamic programming, as in CKY [29]. When a prospective constituent such as "NP from 3 to 8" is built, its *Viterbi inside score* is the log-probability of the best known subparse that matches that description.[1]

A standard extension of the CKY algorithm [12] uses an *agenda*—a priority queue of constituents built so far—to decide which constituent is most promising to extend next, as detailed in section 2.2 below. The success of the inference algorithm in terms of speed and accuracy hinge on its ability to prioritize "good" actions before "bad" actions. In our context, a constituent is "good" if it somehow leads to a high accuracy solution, quickly.

**Running Example 1.** *Either CKY or an agenda-based parser that prioritizes by Viterbi inside score will find the highest-scoring parse. This achieves a percentage accuracy of* 93.3, *given the very large grammar and experimental conditions described in Section 6. However, the agenda-based parser is over an order of magnitude faster than CKY (wall clock time) because it stops as soon as it finds a parse, without building further constituents. With mild pruning according to Viterbi inside score, the accuracy remains* 93.3 *and the speed triples. With more aggressive pruning, the accuracy drops to* 92.0 *and the speed triples again.*

Our goal is to *learn* a prioritization function that satisfies this condition. In order to operationalize this approach, we need to define the test-time objective function we wish to optimize; we choose a simple linear interpolation of accuracy and speed:

$$\text{quality} = \text{accuracy} - \lambda \times \text{time} \tag{1}$$

where we can choose a $\lambda$ that reflects our true preferences. The goal of $\lambda$ is to encode "how much more time am I willing to spend to achieve an additional unit of accuracy?" In this paper, we consider a very simple notion of time: the number of constituents popped from/pushed into the agenda during inference, halting inference as soon as the parser pops its first complete parse.

When considering how to optimize the expectation of Eq (1) over test data, several challenges present themselves. First, this is a *sequential decision process:* the parsing decisions made at a given time may affect both the availability and goodness of future decisions. Second, the parser's total runtime and accuracy on a sentence are unknown until parsing is complete, making this an instance of *delayed reward*. These considerations lead us to formulate this problem as a Markov Decision Process (MDP), a well-studied model of decision processes.

### 2.2 Inference as a Markov Decision Process

A Markov Decision Process (MDP) is a formalization of a memoryless search process. An MDP consists of a *state space* $S$, an *action space* $A$, and a *transition function* $T$. An agent in an MDP observes the current state $s \in S$ and chooses an action $a \in A$. The environment responds by transitioning to a state $s' \in S$, sampled from the transition distribution $T(s' \mid s, a)$. The agent then observes its new state and chooses a new action. An agent's *policy* $\pi$ describes how the (memoryless) agent chooses an action based on its current state, where $\pi$ is either a deterministic function of the state (i.e., $\pi(s) \mapsto a$) or a stochastic distribution over actions (i.e., $\pi(a \mid s)$).

For parsing, the state is the full current chart and agenda (and is astronomically large: roughly $10^{17}$ states for average sentences). The agent controls which item (constituent) to "pop" from the agenda. The initial state has an agenda consisting of all single-word constituents, and an empty *chart* of previously popped constituents. Possible actions correspond to items currently on the agenda. When the agent chooses to pop item $y$, the environment *deterministically* adds $y$ to the chart, combines $y$ as licensed by the grammar with adjacent items $z$ in the chart, and places each resulting new item $x$

on the agenda. (Duplicates in the chart or agenda are merged: the one of highest Viterbi inside score is kept.) The only stochasticity is the initial draw of a new sentence to be parsed.

We are interested in learning a deterministic policy that always pops the highest-priority available action. Thus, learning a policy corresponds to learning a priority function. We define the priority of action $a$ in state $s$ as the dot product of a feature vector $\phi(a, s)$ with the weight vector $\boldsymbol{\theta}$; our features are described in Section 2.3. Formally, our policy is

$$\pi_{\boldsymbol{\theta}}(s) = \arg\max_a \boldsymbol{\theta} \cdot \phi(a, s) \tag{2}$$

An *admissible* policy in the sense of $A^*$ search [13] would guarantee that we always return the parse of highest Viterbi inside score—but we do not require this, instead aiming to optimize Eq (1).

### 2.3   Features for Prioritized Parsing

We use the following simple features to prioritize a possible constituent. (1) Viterbi inside score; (2) constituent touches start of sentence; (3) constituent touches end of sentence; (4) constituent length; (5) $\frac{\text{constituent length}}{\text{sentence length}}$; (6) $\log p(\text{constituent label} \mid \text{prev. word POS tag})$ and $\log p(\text{constituent label} \mid \text{next word POS tag})$, where the part-of-speech (POS) tag of $w$ is taken to be $\arg\max_t p(w \mid t)$ under the grammar; (7) 12 features indicating whether the constituent's {preceding, following, initial} word starts with an {uppercase, lowercase, number, symbol} character; (8) the 5 most positive and 5 most negative punctuation features from [14], which consider the placement of punctuation marks within the constituent.

The log-probability features (1), (6) are inspired by work on figures of merit for agenda-based parsing [4], while case and punctuation patterns (7), (8) are inspired by structure-free parsing [14].

## 3   Reinforcement Learning

Reinforcement learning (RL) provides a generic solution to solving learning problems with delayed reward [25]. The reward function takes a state of the world $s$ and an agent's chosen action $a$ and returns a real value $r$ that indicates the "immediate reward" the agent receives for taking that action. In general the reward function may be stochastic, but in our case, it is deterministic: $r(s, a) \in \mathbb{R}$. The reward function we consider is:

$$r(s, a) = \begin{cases} \text{acc}(a) - \lambda \cdot \text{time}(s) & \text{if } a \text{ is a full parse tree} \\ 0 & \text{otherwise} \end{cases} \tag{3}$$

Here, $\text{acc}(a)$ measures the accuracy of the full parse tree popped by the action $a$ (against a gold standard) and $\text{time}(s)$ is a user-defined measure of time. In words, when the parser completes parsing, it receives reward given by Eq (1); at all other times, it receives no reward.

### 3.1   Boltzmann Exploration

At test time, the transition between states is deterministic: our policy always chooses the action $a$ that has highest priority in the current state $s$. However, during training, we promote exploration of policy space by running with stochastic policies $\pi_{\boldsymbol{\theta}}(a \mid s)$. Thus, there is some chance of popping a lower-priority action, to find out if it is useful and should be given higher-priority. In particular, we use Boltzmann exploration to construct a stochastic policy with a Gibbs distribution. Our policy is:

$$\pi_{\boldsymbol{\theta}}(a \mid s) = \frac{1}{Z(s)} \exp\left[\frac{1}{temp} \boldsymbol{\theta} \cdot \phi(a, s)\right] \text{ with } Z(s) \text{ as the appropriate normalizing constant} \tag{4}$$

That is, the log-likelihood of action $a$ at state $s$ is an affine function of its priority. The temperature *temp* controls the amount of exploration. As $temp \to 0$, $\pi_{\boldsymbol{\theta}}$ approaches the deterministic policy in Eq (2); as $temp \to \infty$, $\pi_{\boldsymbol{\theta}}$ approaches the uniform distribution over available actions. During training, *temp* can be decreased to shift from exploration to exploitation.

A trajectory $\tau$ is the complete sequence of state/action/reward triples from parsing a single sentence. As is common, we denote $\tau = \langle s_0, a_0, r_0, s_1, a_1, r_1, \ldots, s_T, a_T, r_T \rangle$, where: $s_0$ is the starting state; $a_t$ is chosen by the agent by $\pi_{\boldsymbol{\theta}}(a_t \mid s_t)$; $r_t = r(s_t, a_t)$; and $s_{t+1}$ is drawn by the environment from

$T(s_{t+1} \mid s_t, a_t)$, deterministically in our case. At a given temperature, the weight vector $\boldsymbol{\theta}$ gives rise to a distribution over trajectories and hence to an expected total reward:

$$R = \mathbb{E}_{\tau \sim \pi_{\boldsymbol{\theta}}}\left[R(\tau)\right] = \mathbb{E}_{\tau \sim \pi_{\boldsymbol{\theta}}}\left[\sum_{t=0}^{T} r_t\right]. \tag{5}$$

where $\tau$ is a random trajectory chosen by policy $\pi_{\boldsymbol{\theta}}$, and $r_t$ is the reward at step $t$ of $\tau$.

## 3.2 Policy Gradient

Given our features, we wish to find parameters that yield the highest possible expected reward. We carry out this optimization using a stochastic gradient ascent algorithm known as policy gradient [27, 26]. This operates by taking steps in the direction of $\nabla_{\boldsymbol{\theta}} R$:

$$\nabla_{\boldsymbol{\theta}} \mathbb{E}_{\tau}[R(\tau)] = \mathbb{E}_{\tau}\left[\frac{\nabla_{\boldsymbol{\theta}} p_{\boldsymbol{\theta}}(\tau)}{p_{\boldsymbol{\theta}}(\tau)} R(\tau)\right] = \mathbb{E}_{\tau}\left[R(\tau) \nabla_{\boldsymbol{\theta}} \log p_{\boldsymbol{\theta}}(\tau)\right] = \mathbb{E}_{\tau}\left[R(\tau) \sum_{t=0}^{T} \nabla_{\boldsymbol{\theta}} \log \pi(a_t \mid s_t)\right] \tag{6}$$

The expectation can be approximated by sampling trajectories. It also requires computing the gradient of each policy decision, which, by Eq (4), is:

$$\nabla_{\boldsymbol{\theta}} \log \pi_{\boldsymbol{\theta}}(a_t \mid s_t) = \frac{1}{temp}\left(\phi(a_t, s_t) - \sum_{a' \in A} \pi_{\boldsymbol{\theta}}(a' \mid s_t)\phi(a', s_t)\right) \tag{7}$$

Combining Eq (6) and Eq (7) gives the form of the gradient with respect to a single trajectory. The policy gradient algorithm samples one trajectory (or several) according to the current $\pi_{\boldsymbol{\theta}}$, and then takes a gradient step according to Eq (6). This increases the probability of actions on high-reward trajectories more than actions on low-reward trajectories.

**Running Example 2.** *The baseline system from Running Example 1 always returns the target parse (the complete parse with maximum Viterbi inside score). This achieves an accuracy of 93.3 (percent recall) and speed of* 1.5 *mpops (million pops) on training data. Unfortunately, running policy gradient from this starting point degrades speed* and *accuracy. Training is not practically feasible: even the first pass over 100 training sentences (sampling 5 trajectories per sentence) takes over a day.*

## 3.3 Analysis

One might wonder *why* policy gradient performed so poorly on this problem. One hypothesis is that it is the fault of stochastic gradient descent: the optimization problem was too hard or our step sizes were chosen poorly. To address this, we attempted an experiment where we added a "cheating" feature to the model, which had a value of one for constituents that should be in the final parse, and zero otherwise. Under almost every condition, policy gradient was able to learn a near-optimal policy by placing high weight on this cheating feature.

An alternative hypothesis is overfitting to the training data. However, we were unable to achieve significantly higher accuracy even when evaluating on our training data—indeed, even for a single train/test sentence.

The main difficulty with policy gradient is *credit assignment*: it has no way to determine which actions were "responsible" for a trajectory's reward. Without causal reasoning, we need to sample many trajectories in order to distinguish which actions are reliably associated with higher-reward. This is a significant problem for us, since the average trajectory length of an $A_0^*$ parser on a $15$ word sentence is about 30,000 steps, only about 40 of which (less than $0.15\%$) are actually needed to successfully complete the parse optimally.

## 3.4 Reward Shaping

A classic approach to attenuating the credit assignment problem when one has some knowledge about the domain is *reward shaping* [10]. The goal of reward shaping is to heuristically associate portions of the total reward with specific time steps, and to favor actions that are observed to be soon followed by a reward, on the assumption that they caused that reward.

If speed is measured by the number of popped items and accuracy is measured by labeled constituent recall of the first-popped complete parse (compared to the gold-standard parse), one natural way to shape rewards is to give an immediate penalty for the time incurred in performing the action while giving an immediate positive reward for actions that build constituents of the gold parse. Since only some of the correct constituents built may actually make it into the returned tree, we can correct for having "incorrectly" rewarded the others by penalizing the final action. Thus, the shaped reward:

$$\tilde{r}(s,a) = \begin{cases} 1 - \Delta(s,a) - \lambda & \text{if } a \text{ pops a complete parse (causing the parser to halt and return } a) \\ 1 - \lambda & \text{if } a \text{ pops a labeled constituent that appears in the gold parse} \\ -\lambda & \text{otherwise} \end{cases} \tag{8}$$

$\lambda$ is from Eq (1), penalizing the runtime of each step. 1 rewards a correct constituent. The correction $\Delta(s,a)$ is the number of correct constituents popped into the chart of $s$ that were not in the first-popped parse $a$. It is easy to see that for any trajectory ending in a complete parse, the total shaped and unshaped rewards along a trajectory are equal (i.e. $r(\tau) = \tilde{r}(\tau)$).

We now modify the total reward to use temporal discounting. Let $0 \leq \gamma \leq 1$ be a discount factor. When rewards are discounted over time, the policy gradient becomes

$$\mathbb{E}_{\tau \sim \pi_{\boldsymbol{\theta}}}[\tilde{R}_\gamma(\tau)] = \mathbb{E}_{\tau \sim \pi_{\boldsymbol{\theta}}}\left[\sum_{t=0}^{T}\left(\sum_{t'=t}^{T}\gamma^{t'-t}\tilde{r}_{t'}\right)\nabla_{\boldsymbol{\theta}}\log \pi_{\boldsymbol{\theta}}(a_t \mid s_t)\right] \tag{9}$$

where $\tilde{r}_{t'} = \tilde{r}(s_{t'}, a_{t'})$. When $\gamma = 1$, the gradient of the above turns out to be equivalent to Eq (6) [20, section 3.1], and therefore following the gradient is equivalent to policy gradient. When $\gamma = 0$, the parser gets only immediate reward—and in general, a small $\gamma$ assigns the credit for a local reward $\tilde{r}_{t'}$ mainly to actions $a_t$ at closely preceding times.

This gradient step can now achieve some credit assignment. If an action is on a good trajectory but occurs *after* most of the useful actions (pops of correct constituents), then it does *not* receive credit for those previously occurring actions. However, if it occurs *before* useful actions, it still does receive credit because we do not know (without additional simulation) whether it was a necessary step toward those actions.

**Running Example 3.** *Reward shaping helps significantly, but not enough to be competitive. As the parser speeds up, training is about 10 times faster than before. The best setting ($\gamma = 0, \lambda = 10^{-6}$) achieves an accuracy in the mid-70's with only about $0.2$ mpops. No settings were able to achieve higher accuracy.*

# 4 Apprenticeship Learning

In reinforcement learning, an agent interacts with an environment and attempts to learn to maximize its reward by repeating actions that led to high reward in the past. In apprenticeship learning, we assume access to a collection of trajectories taken by an *optimal policy* and attempt to learn to mimic those trajectories. The learner's only goal is to behave like the teacher at every step: it does not have any notion of reward. In contrast, the related task of inverse reinforcement learning/optimal control [17, 11] attempts to infer a reward function from the teacher's optimal behavior.

Many algorithms exist for apprenticeship learning. Some of them work by first executing inverse reinforcement learning [11, 17] to induce a reward function and then feeding this reward function into an off-the-shelf reinforcement learning algorithm like policy gradient to learn an approximately optimal agent [1]. Alternatively, one can directly learn to mimic an optimal demonstrator, without going through the side task of trying to induce its reward function [7, 24].

## 4.1 Oracle Actions

With a teacher to help guide the learning process, we would like to explore more intelligently than Boltzmann exploration, in particular, focusing on high-reward regions of policy space. We introduce *oracle actions* as a guidance for areas to explore.

Ideally, oracle actions should lead to a maximum-reward tree. In training, we will identify oracle actions to be those that build items in the maximum likelihood parse consistent with the gold parse. When multiple oracle actions are available on the agenda, we will break ties according to the priority assigned by the current policy (i.e., choose the oracle action that it currently likes best).

## 4.2 Apprenticeship Learning via Classification

Given a notion of oracle actions, a straightforward approach to policy learning is to simply train a classifier to follow the oracle—a popular approach in incremental parsing [6, 5]. Indeed, this serves as the initial iteration of the state-of-the-art apprenticeship learning algorithm, DAGGER [24].

We train a classifier as follows. Trajectories are generated by following oracle actions, breaking ties using the initial policy (Viterbi inside score) when multiple oracle actions are available. These trajectories are incredibly

short (roughly double the number of words in the sentence). At each step in the trajectory, $(s_t, a_t)$, a classification example is generated, where the action taken by the oracle $(a_t)$ is considered the correct class and all other available actions are considered incorrect. The classifier that we train on these examples is a maximum entropy classifier, so it has *exactly* the same form as the Boltzmann exploration model (Eq (4)) but without the temperature control. In fact, the *gradient* of this classifier (Eq (10)) is nearly identical to the policy gradient (Eq (6)) except that $\tau$ is distributed differently and the total reward $R(\tau)$ does not appear: instead of mimicking high-reward trajectories we now try to mimic oracle trajectories.

$$\mathbb{E}_{\tau \sim \pi^*} \left[ \sum_{t=0}^{T} \left( \phi(a_t, s_t) - \sum_{a' \in A} \pi_{\boldsymbol{\theta}}(a' \mid s_t) \phi(a', s_t) \right) \right] \tag{10}$$

where $\pi^*$ denotes the oracle policy so $a_t$ is the oracle action. The potential benefit of the classifier-based approach over policy gradient with shaped rewards is increased credit assignment. In policy gradient with reward shaping, an action gets credit for all future reward (though no past reward). In the classifier-based approach, it gets credit for exactly whether or not it builds an item that is in the true parse.

**Running Example 4.** *The classifier-based approach performs only marginally better than policy gradient with shaped rewards. The best accuracy we can obtain is* 76.5 *with* 0.19 *mpops.*

To execute the DAGGER algorithm, we would continue in the next iteration by following the trajectories learned by the classifier and generating new classification examples on those states. Unfortunately, this is not computationally feasible due to the poor quality of the policy learned in the first iteration. Attempting to follow the learned policy essentially tries to build *all* possible constituents licensed by the grammar, which can be prohibitively expensive. We will remedy this in section 5.

### 4.3  What's Wrong With Apprenticeship Learning

An obvious practical issue with the classifier-based approach is that it trains the classifier only at states visited by the oracle. This leads to the well-known problem that it is unable to learn to recover from past errors [2, 28, 7, 24]. Even though our current feature set depends *only* on the action and not on the state, making action scores independent of the current state, there is still an issue since the set of actions to choose from does depend on the state. That is, the classifier is trained to discriminate only among the small set of agenda items available on the oracle trajectory (which are always combinations of *correct* constituents). But the action sets the parser faces at test time are much larger and more diverse.

An additional objection to classifiers is that not all errors are created equal. Some incorrect actions are more expensive than others, if they create constituents that can be combined in many locally-attractive ways and hence slow the parser down or result in errors. Our classification problem does not distinguish among incorrect actions. The SEARN algorithm [7] would distinguish them by explicitly evaluating the future reward of each possible action (instead of using a teacher) and incorporating this into the classification problem. But explicit evaluation is computationally infeasible in our setting (at each time step, it must roll out a full future trajectory for *each* possible action from the agenda). Policy gradient provides another approach by observing which actions are good or bad across many random trajectories, but recall that we found it impractical as well. We do not further address this problem in this paper, but in [8] we suggested explicit causality analysis.

A final issue has to do with the nature of the oracle. Recall that the oracle is "supposed to" choose optimal actions for the given reward. Also recall that our oracle always picks correct constituents. There seems to be a contradiction here: our oracle action selector *ignores* $\lambda$, the tradeoff between accuracy and speed, and only focuses on accuracy. This happens because for any reasonable setting of $\lambda$, the optimal thing to do is always to just build the correct tree without building any extra constituents. Only for *very* large values of $\lambda$ is it optimal to do anything else, and for such values of $\lambda$, the learned model will have hugely negative reward. This means that under the apprenticeship learning setting, we are actually *never* going to be able to learn to trade off accuracy and speed: as far as the oracle is concerned, you can have both! The tradeoff only appears because our model cannot come remotely close to mimicking the oracle.

## 5  Oracle-Infused Policy Gradient

The failure of both standard reinforcement learning algorithms and standard apprenticeship learning algorithms on our problem leads us to develop a new approach. We start with the policy gradient algorithm (Section 3.2) and use ideas from apprenticeship learning to improve it. Our formulation preserves the reinforcement learning flavor of our overall setting, which involves delayed reward for a *known* reward function.

Our approach is specifically designed for the non-deterministic nature of the agenda-based parsing setting [8]: once some action $a$ becomes available (appears on the agenda), it never goes away until it is taken. This makes the notion of "interleaving" oracle actions with policy actions both feasible and sensible. Like policy gradient, we draw trajectories from a policy and take gradient steps that favor actions with high reward under reward shaping. Like SEARN and DAGGER, we begin by exploring the space around the optimal policy and slowly explore out from there.

To achieve this, we define the notion of an *oracle-infused policy*. Let $\pi$ be an arbitrary policy and let $\delta \in [0, 1]$. We define the oracle-infused policy $\pi_\delta^+$ as follows:

$$\pi_\delta^+(a \mid s) = \delta\pi^*(a \mid s) + (1 - \delta)\pi(a \mid s) \tag{11}$$

In other words, when choosing an action, $\pi_\delta^+$ explores the policy space with probability $1 - \delta$ (according to its current model), but with probability $\delta$, we force it to take an oracle action.

Our algorithm takes policy gradient steps with reward shaping (Eqs (9) and (7)), but with respect to trajectories drawn from $\pi_\delta^+$ rather than $\pi$. If $\delta = 0$, it reduces to policy gradient, with reward shaping if $\gamma < 1$ and immediate reward if $\gamma = 0$. For $\delta = 1$, the $\gamma = 0$ case reduces to the classifier-based approach with $\pi^*$ (which in turn breaks ties by choosing the best action under $\pi$).

Similar to DAGGER and SEARN, we do not stay at $\delta = 1$, but wean our learner off the oracle supervision as it starts to find a good policy $\pi$ that imitates the classifier reasonably well. We use $\delta = 0.8^{\text{epoch}}$, where epoch is the total number of passes made through the training set at that point (so $\delta = 0.8^0 = 1$ on the initial pass). Over time, $\delta \rightarrow 0$, so that eventually we are training the policy to do well on the same distribution of states that it will pass through at test time (as in policy gradient). With intermediate values of $\delta$ (and $\gamma \approx 1$), an iteration behaves similarly to an iteration of SEARN, except that it "rolls out" the consequences of an action chosen randomly from (11) instead of evaluating all possible actions in parallel.

**Running Example 5.** *Oracle-infusion gives a competitive speed and accuracy tradeoff. A typical result is* $91.2$ *with* $0.68$ *mpops.*

# 6 Experiments

All of our experiments (including those discussed earlier) are based on the Wall Street Journal portion of the Penn Treebank [15]. We use a probabilistic context-free grammar with 370,396 rules—enough to make the baseline system accurate but slow. We obtained it as a latent-variable grammar [16] using 5 split-merge iterations [21] on sections 2–20 of the Treebank, reserving section 22 for learning the parameters of our policy. All approaches to trading off speed and accuracy are trained on section 22; in particular, for the running example and Section 6.2, the same 100 sentences of at most 15 words from that section were used for training and test. We measure accuracy in terms of labeled recall (including preterminals) and measure speed in terms of the number of pops from on the agenda. The limitation to relatively short sentences is purely for improved efficiency at training time.

## 6.1 Baseline Approaches

Our baseline approaches trade off speed and accuracy not by learning to prioritize, but by varying the pruning level $\Delta$. A constituent is pruned if its Viterbi inside score is more than $\Delta$ worse than that of some other constituent that covers the same substring.

Our baselines are: **(HA$^*$)** a Hierarchical A$^*$ parser [18] with the same pruning threshold at each hierarchy level; **(A$_0^*$)** an A$^*$ parser with a 0 heuristic function plus pruning; **(IDA$_0^*$)** an iterative deepening A$^*$ algorithm, on which a failure to find any parse causes us to increase $\Delta$ and try again with less aggressive pruning (note that this is not the traditional meaning of IDA*); and **(CTF)** the default coarse-to-fine parser in the Berkeley parser [21]. Several of these algorithms can make multiple passes, in which case the runtime (number of pops) is assessed *cumulatively*.

## 6.2 Learned Prioritization Approaches

We explored four variants of our oracle-infused policy gradient with with $\lambda = 10^{-6}$. Figure 1 shows the result on the 100 training sentences. The "-" tests are the degenerate case of $\delta = 1$, or apprenticeship learning (section 4.2), while the "+" tests use $\delta = 0.8^{\text{epoch}}$ as recommended in section 5. Temperature matters for the "+" tests and we use $temp = 1$. We performed stochastic gradient descent for 25 passes over the data, sampling 5 trajectories in a row for each sentence (when $\delta < 1$ so that trajectories are random).

| Model | # of pops | Recall | F1 |
|---|---|---|---|
| A$_0^*$ (no pruning) | 1496080 | 93.34 | 93.19 |
| D− | 686641 | 56.35 | 58.74 |
| I− | 187403 | 76.48 | 76.92 |
| D+ | 1275292 | 84.17 | 83.38 |
| I+ | 682540 | 91.16 | 91.33 |

Figure 1: Performance on 100 sentences.

We can see that the classifier-based approaches "-" perform poorly: when training trajectories consist of *only* oracle actions, learning is severely biased. Yet we saw in section 3.2 that without *any* help from the oracle actions, we suffer from such large variance in the training trajectories that performance degrades rapidly and learning does not converge even after days of training. Our "oracle-infused" compromise "+" uses *some* oracle actions: after several passes through the data, the parser learns to make good decisions without help from the oracle.

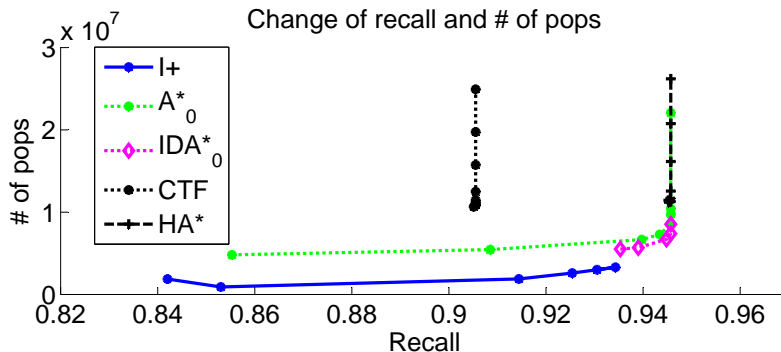

Figure 2: Pareto frontiers: Our `I+` parser at different values of $\lambda$, against the baselines at different pruning levels.

The other axis of variation is that the "**D**" tests (delayed reward) use $\gamma = 1$, while the "**I**" tests (immediate reward) use $\gamma = 0$. Note that **I+** attempts a form of credit assignment and works better than **D+**.[2] We were not able to get better results with intermediate values of $\gamma$, presumably because this crudely assigns credit for a reward (correct constituent) to the actions that closely preceded it, whereas in our agenda-based parser, the causes of the reward (correct subconstituents) related actions may have happened much earlier [8].

### 6.3 Pareto Frontier

Our final evaluation is on the held-out test set (length-limited sentences from Section 23). A 5-split grammar trained on section 2-21 is used. Given our previous results in Table 1, we only consider the **I+** model: immediate reward with oracle infusion. To investigate trading off speed and accuracy, we learn and then evaluate a policy for each of several settings of the tradeoff parameter: $\lambda$. We train our policy using sentences of at most 15 words from Section 22 and evaluate the learned policy on the held out data (from Section 23). We measure accuracy as labeled constituent recall and evaluate speed in terms of the number of pops (or pushes) performed on the agenda.

Figure 2 shows the baselines at different pruning thresholds as well as the performance of our policies trained using **I+** for $\lambda \in \{10^{-3}, 10^{-4}, \ldots, 10^{-8}\}$, using agenda pops as the measure of time. **I+** is about 3 times as fast as unpruned $\mathbf{A}_0^*$ at the cost of about 1% drop in accuracy (F-score from 94.58 to 93.56). Thus, **I+** achieves the same accuracy as the pruned version of $\mathbf{A}_0^*$ while still being twice as fast. **I+** also improves upon $\mathbf{HA}^*$ and $\mathbf{IDA}_0^*$ with respect to speed at 60% of the pops. **I+** always does better than the coarse-to-fine parser (**CTF**) in terms of **both** speed and accuracy, though using the number of agenda pops as our measure of speed puts both of our hierarchical baselines at a disadvantage.

We also ran experiments using the number of agenda *pushes* as a more accurate measure of time, again sweeping over settings of $\lambda$. Since our reward shaping was crafted with agenda pops in mind, perhaps it is not surprising that learning performs relatively poorly in this setting. Still, we do manage to learn to trade off speed and accuracy. With a 1% drop in recall (F-score from 94.58 to 93.54), we speed up from $A_0^*$ by a factor of 4 (from around 8 billion pushes to 2 billion). Note that known pruning methods could also be employed in conjunction with learned prioritization.

## 7 Conclusions and Future Work

In this paper, we considered the application of both reinforcement learning and apprenticeship learning to prioritize search in a way that is sensitive to a user-defined tradeoff between speed and accuracy. We found that a novel oracle-infused variant of the policy gradient algorithm for reinforcement learning is effective for learning a fast and accurate parser with only a simple set of features. In addition, we uncovered many properties of this problem that separate it from more standard learning scenarios, and designed experiments to determine the *reasons* off-the-shelf learning algorithms fail.

An important avenue for future work is to consider better credit assignment. We are also very interested in designing richer feature sets, including "dynamic" features that depend on both the action *and* the state of the chart and agenda. One role for dynamic features is to decide when to halt. The parser might decide to continue working past the first complete parse, or give up (returning a partial or default parse) before any complete parse is found.

## Footnotes

[1]E.g., the maximum log-probability of generating some tree whose fringe is the substring spanning words (3,8], given that NP (noun phrase) is the root nonterminal. This is the total log-probability of rules in the tree.

[2]The **D-** and **I-** approaches are quite similar to each other. Both train on oracle trajectories where all actions receive a reward of $1 - \lambda$, and simply try to make these oracle actions probable. However, **D-** trains more aggressively on long trajectories, since (9) implies that it weights a given training action by $T - t + 1$, the number of future actions on that trajectory. The difference between **D+** and **I+** is more interesting because the trajectory includes non-oracle actions as well.

# References

[1] Pieter Abbeel and Andrew Ng. Apprenticeship learning via inverse reinforcement learning. In *ICML*, 2004.

[2] J. Andrew Bagnell. Robust supervised learning. In *AAAI*, 2005.

[3] Nathan Bodenstab, Aaron Dunlop, Keith Hall, and Brian Roark. Beam-width prediction for efficient CYK parsing. In *ACL*, 2011.

[4] Sharon A. Caraballo and Eugene Charniak. New figures of merit for best-first probabilistic chart parsing. *Computational Linguistics*, 24(2):275–298, 1998.

[5] Eugene Charniak. Top-down nearly-context-sensitive parsing. In *EMNLP*, 2010.

[6] Michael Collins and Brian Roark. Incremental parsing with the perceptron algorithm. In *ACL*, 2004.

[7] Hal Daumé III, John Langford, and Daniel Marcu. Search-based structured prediction. *Machine Learning*, 75(3):297–325, 2009.

[8] Jason Eisner and Hal Daumé III. Learning speed-accuracy tradeoffs in nondeterministic inference algorithms. In *COST: NIPS Workshop on Computational Trade-offs in Statistical Learning*, 2011.

[9] Joshua Goodman. Semiring parsing. *Computational Linguistics*, 25(4):573–605, December 1999.

[10] V. Gullapalli and A. G. Barto. Shaping as a method for accelerating reinforcement learning. In *Proceedings of the IEEE International Symposium on Intelligent Control*, 1992.

[11] R. Kalman. Contributions to the theory of optimal control. *Bol. Soc. Mat. Mexicana*, 5:558–563, 1968.

[12] Martin Kay. Algorithm schemata and data structures in syntactic processing. In B. J. Grosz, K. Sparck Jones, and B. L. Webber, editors, *Readings in Natural Language Processing*, pages 35–70. Kaufmann, 1986. First published (1980) as Xerox PARC TR CSL-80-12.

[13] Dan Klein and Chris Manning. A* parsing: Fast exact Viterbi parse selection. In *NAACL/HLT*, 2003.

[14] Percy Liang, Hal Daumé III, and Dan Klein. Structure compilation: Trading structure for features. In *ICML*, Helsinki, Finland, 2008.

[15] M.P. Marcus, M.A. Marcinkiewicz, and B. Santorini. Building a large annotated corpus of English: The Penn Treebank. *Computational linguistics*, 19(2):330, 1993.

[16] Takuya Matsuzaki, Yusuke Miyao, and Junichi Tsujii. Probabilistic CFG with latent annotations. In *ACL*, 2005.

[17] Andrew Ng and Stuart Russell. Algorithms for inverse reinforcement learning. In *ICML*, 2000.

[18] A. Pauls and D. Klein. Hierarchical search for parsing. In *NAACL/HLT*, pages 557–565. Association for Computational Linguistics, 2009.

[19] A. Pauls and D. Klein. Hierarchical A* parsing with bridge outside scores. In *ACL*, pages 348–352. Association for Computational Linguistics, 2010.

[20] Jan Peters and Stefan Schaal. Reinforcement learning of motor skills with policy gradients. *Neural Networks*, 21(4), 2008.

[21] S. Petrov and D. Klein. Improved inference for unlexicalized parsing. In *NAACL/HLT*, pages 404–411, 2007.

[22] B. Roark, K. Hollingshead, and N. Bodenstab. Finite-state chart constraints for reduced complexity context-free parsing pipelines. *Computational Linguistics*, Early Access:1–35, 2012.

[23] Brian Roark and Kristy Hollingshead. Classifying chart cells for quadratic complexity context-free inference. In *COLING*, pages 745–752, Manchester, UK, August 2008. Coling 2008 Organizing Committee.

[24] Stephane Ross, Geoff J. Gordon, and J. Andrew Bagnell. A reduction of imitation learning and structured prediction to no-regret online learning. In *AI-Stats*, 2011.

[25] Richard Sutton and Andrew Barto. *Reinforcement Learning: An Introduction*. MIT Press, 1998.

[26] Richard S. Sutton, David McAllester, Satinder Singh, and Yishay Mansour. Policy gradient methods for reinforcement learning with function approximation. In *NIPS*, pages 1057–1063. MIT Press, 2000.

[27] R.J. Williams. Simple statistical gradient-following algorithms for connectionist reinforcement learning. *Machine Learning*, 8(23), 1992.

[28] Yuehua Xu and Alan Fern. On learning linear ranking functions for beam search. In *ICML*, pages 1047–1054, 2007.

[29] D. H. Younger. Recognition and parsing of context-free languages in time $n^3$. *Information and Control*, 10(2):189–208, February 1967.

